# Online Models for Content Optimization

**Deepak Agarwal, Bee-Chung Chen, Pradheep Elango, Nitin Motgi, Seung-Taek Park,
Raghu Ramakrishnan, Scott Roy, Joe Zachariah**

Yahoo! Inc.

701 First Avenue

Sunnyvale, CA 94089

## Abstract

We describe a new content publishing system that selects articles to serve to a user, choosing from an editorially programmed pool that is frequently refreshed. It is now deployed on a major Yahoo! portal, and selects articles to serve to hundreds of millions of user visits per day, significantly increasing the number of user clicks over the original manual approach, in which editors periodically selected articles to display. Some of the challenges we face include a dynamic content pool, short article lifetimes, non-stationary click-through rates, and extremely high traffic volumes. The fundamental problem we must solve is to quickly identify which items are popular (perhaps within different user segments), and to exploit them while they remain current. We must also explore the underlying pool constantly to identify promising alternatives, quickly discarding poor performers. Our approach is based on tracking per article performance in near real time through online models. We describe the characteristics and constraints of our application setting, discuss our design choices, and show the importance and effectiveness of coupling online models with a randomization procedure. We discuss the challenges encountered in a production online content-publishing environment and highlight issues that deserve careful attention. Our analysis of this application also suggests a number of future research avenues.

## 1  Introduction

The web has become the central distribution channel for information from traditional sources such as news outlets as well as rapidly growing user-generated content. Developing effective algorithmic approaches to delivering such content when users visit web portals is a fundamental problem that has not received much attention. Search engines use automated ranking algorithms to return the most relevant links in response to a user's keyword query; likewise, online ads are targeted using automated algorithms. In contrast, portals that cater to users who browse a site are typically programmed manually. This is because content is harder to assess for relevance, topicality, freshness, and personal preference; there is a wide range in the quality; and there are no reliable quality or trust metrics (such as, say, PageRank or Hub/Authority weights for URLs).

Manual programming of content ensures high quality and maintains the editorial "voice" (the typical mix of content) that users associate with the site. On the other hand, it is expensive to scale as the number of articles and the number of site pages we wish to program grow. A data-driven machine learning approach can help with the scale issue, and we seek to blend the strengths of the editorial and algorithmic approaches by algorithmically optimizing content programming within high-level constraints set by editors. The system we describe is currently deployed on a major Yahoo! portal, and serves several hundred million user visits per day.

The usual machine-learning approach to ranking articles shown to users uses feature-based models, trained using "offline data" (data collected in the past). After making a significant effort of feature

engineering by looking at user demogrpahics, past activities on the site, various article categories, keywords and entities in articles, etc., we concluded that it is difficult to build good models based solely on offline data in our scenario. Our content pool is small but changing rapidly; article lifetimes are short; and there is wide variability in article performance sharing a common set of feature values. Thus, we take the approach of tracking per-article performance by online models, which are initialized using offline data and updated continuously using real time data. This online aspect opens up new modeling challenges in addition to classical feature based predition, as we discuss in this paper.

## 2 Problem Description

We consider the problem of optimizing content displayed in a module that is the focal point on a major Yahoo! portal; the page also provides several other services (e.g., Mail, Weather) and content links. The module is a panel with four slots labelled F1, F2, F3, F4. Slot F1, which accounts for a large fraction of clicks, is prominent, and an article displayed on F1 receives many more clicks than when it is displayed at F2, F3 or F4.

The pool of available articles is created by trained human editors, and refreshed continually. At any point in time, there are 16 live articles in the pool. A few new articles programmed by editors get *pushed* into the system periodically (every few hours) and replace some old articles. The editors keep up with important new stories (e.g., breaking news) and eliminate irrelevant and fading stories, and ensure that the pool of articles is consistent with the "voice" of the site (i.e., the desired nature and mix of content). There is no personalization in the editorially programmed system; at a given time, the same articles are seen by all users visiting the page.

We consider how to choose the best set of four articles to display on the module to a given user. Since the mix of content in the available pool already incorporates constraints like *voice*, topicality, etc., we focus on choosing articles to maximize overall click-through rate (CTR), which is the total number of clicks divided by total number of views for a time interval. To simplify our presentation, we only describe learning from click feedback obtained from the most important F1 position; our framework (and system) can use information from other positions as well.

### 2.1 System Challenges

Our setting poses many challenges, the most important of which are the following:

- **Highly dynamic system characteristics**: Articles have short lifetimes (6-8 hours), the pool of available articles is constantly changing, the user population is dynamic, and each article has different CTRs at different times of day or when shown in different slots in our module. We found that fast reaction to user feedback through dynamic models based on clicks and page views is crucial for good performance. We discuss an alternate and commonly pursued approach of ranking articles based on offline feature-driven models in Section 2.2.

- **Scalability**: The portal receives thousands of page views per second and serves hundreds of millions of user visits per day. Data collection, model training and article scoring (using the model) are subject to tight latency requirements. For instance, we only get a few milliseconds to decide on the appropriate content to show to a user visiting the portal.

A significant effort was required to build a scalable infrastructure that supports near real-time data collection.[1] Events (users' clicks and page views) are collected from a large number of front-end web servers and continuously transferred to data collection clusters, which support event buffering to handle the time lag between the user viewing a page and then clicking articles on the page. The event stream is then fed to the modeler cluster which runs learning algorithms to update the models. Periodically, the front-end web servers pull the updated models and serve content based on the new models. A complete cycle of data collection, model update, and model delivery takes a few minutes.

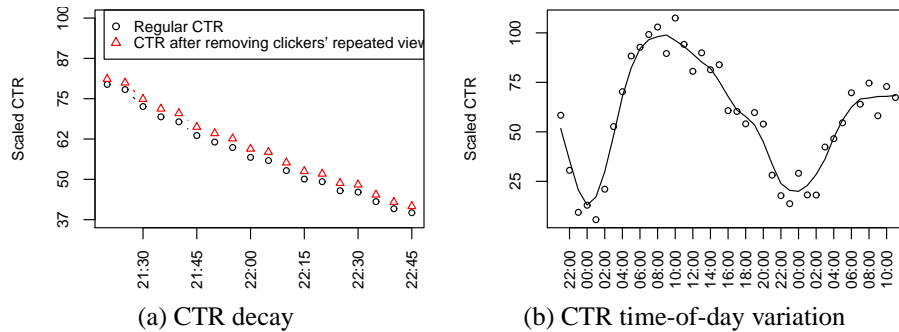

|  (a) CTR decay | (b) CTR time-of-day variation |

Figure 1: *CTR curves of a typical article in two buckets. (a) shows the article's CTR decay when shown continuously in a bucket at position F1; (b) shows the article's CTR in the random bucket.*

## 2.2 Machine Learning Challenges

A *serving scheme* is an automated or manual algorithm that decides which article to show at different positions of our module for a given user. Prior to our system, articles were chosen by human editors; we refer to this as the *editorial* serving scheme. A random sample of the user population is referred to as a *bucket*.

We now discuss the issues that make it tricky to build predictive models in this setting. We tried the usual approach of building offline models based on retrospective data collected while using the editorial serving scheme. User features included Age, Gender, Geo-location and Inferred interests based on user visit patterns. For articles, we used features based on URL, article category (e.g., Sports, Entertainment) and title keywords. However, this approach performed poorly. The reasons include a wide variability in CTR for articles having a same set of feature values, dramatical changes of article CTR over time, and the fact that retrospective data collected from non-randomized serving schemes are confounded with factors that are hard to adjust for (see Section 5). Also, our initial studies revealed high variability in CTRs for articles sharing some common features (e.g., Sports articles, Entertainment articles). We achieved much better performance by seeking quick convergence (using online models) to the best article for a given user (or user segment); a lost opportunity (failure to detect the best article quickly) can be costly and the cost increases with the *margin* (difference between the best and selected articles). We now discuss some of the challenges we had to address.

**Non-stationary CTRs**: The CTR of an article is strongly dependent on the serving scheme used (especially, how much F1 exposure it receives) and it may change dramatically over time. Hence, learning techniques that assume process stationarity are inapplicable. In order to ensure webpage stability, we consider serving schemes that don't alter the choice of what to show a user in a given slot until a better choice is identified. Figure 1 (a) shows the CTR curve of a typical article subject to such a serving scheme. The decay is due to users getting *exposed* to an article more than once. Exposure to an article happens in different ways and to different degrees. A user may get exposed to an article when he/she sees a descriptive link, or clicks on it and reads the article. A user may also click multiple "see also" links associated with each article which may perhaps be a stronger form of exposure. In our analysis, we consider such related clicks to be a single click event. View exposure is more noisy since our module is only one of many content pieces shown on the portal. A user may be looking at the Weather module when visiting the portal or he may have looked at the article title in the link and not liked it. Hence, explaining the decay precisely in terms of repeat exposure is difficult. For instance, not showing an article to a user after one page view containing the link may be suboptimal since he may have overlooked the link and may click on it later. In fact, a large number of clicks on articles occur after the first page view and depends on how a user navigates the portal. Instead of solving the problem by imposing serving constraints per user, we build a component in our dynamic model that tracks article CTR decay over time. We still impose reasonable serving constraints to provide good user experience—we do not show the same article to a user $x$ minutes ($x = 60$ worked well) after he/she first saw the article.

In addition to decay, the CTR of an article also changes by time of day and day of week. Figure 1 (b) shows the CTR of a typical article when served using a randomized serving scheme (articles served in a round-robin fashion to a randomly chosen user population). The randomization removes any serving bias and provides an unbiased estimate of CTR seasonality. It is evident that CTRs of articles vary dramatically over time, this clearly shows the need to adjust for time effects (e.g.,

diurnal patterns, decay) to obtain an adjusted article score when deciding to rank articles. In our current study, we fitted a global time of day curve at 5 minute resolution to data obtained from randomized serving scheme through a periodic (weekly) adaptive regression spline. However, there are still interactions that occur at an article level which were difficult to estimate offline through article features. Per-article online models that put more weight on recent observations provide an effective self adaptive mechanism to automatically account for deviations from the global trend when an article is pushed into the system.

**Strong Serving Bias**: A model built using data generated from a serving scheme is biased by that scheme. For example, if a serving scheme decides not to show article $A$ to any user, any model built using this data would not learn the popularity of $A$ from users' feedback. In general, a serving scheme may heavily exploit some regions in the feature space and generate many data points for those regions, but few data points for other regions. Models built using such data learn very little about the infrequently sampled regions. Moreover, every non-randomized serving scheme introduces confounding factors in the data; adjusting such factors to obtain unbiased article scores is often difficult. In fact, early experiments that updated models using data from editorial bucket to serve in our experimental buckets performed poorly. This bias also affects empirical evaluations or comparisons of learning algorithms based on retrospective data, as we discuss later in Section 5.

**Interaction with the editorial team**: The project involved considerable interaction with human editors who have been manually and successfully placing articles on the portal for many years. Understanding how that experience can be leveraged in conjuction with automated serving schemes was a major challenge, both technically and culturally (in that editors had to learn what ML algorithms could do, and we had to learn all the subtle considerations in what to show). The result is our framework, wherein editors control the pool and set policies via constraints on what can be served, and the serving algorithm chooses what to show on a given user visit.

## 3 Experimental Setup and Framework

**Experimental Setup**: We created several mutually exclusive *buckets* of roughly equal sizes from a fraction of live traffic, and served traffic in each bucket using one of our candidate serving schemes. All usual precautions were taken in the bucket creation process to ensure statistical validity of results. We also created a control bucket that ran the editorial serving scheme. In addition, we created a separate bucket called the *random* bucket, which serves articles per visit in a round-robin fashion.

**Framework**: Our framework consists of several components that are described below.

- **Batch Learning**: Due to time lag between a view and subsequent clicks (approximately 2-10 minutes) and engineering constraints imposed by high data volumes, updates to our models occur every 5 minutes. Such constraints can be added by editors, and the serving algorithm must satisfy them.

- **Business Logic, Editorial overrides**: Despite moving towards an algorithmic approach, there are instances where the editorial team has to override the recommendations produced by our machine learning algorithms. For instance, a breaking news story is shown immediately at the F1 position, a user visiting the portal after 60 minutes should not see the same article he saw during his earlier visits, etc. Such constraints can be added by editors, and the serving algo must satisfy them.

- **Online models to track CTR**: We build online models to track article CTR for various user segments separately in each bucket. Articles that are currently the best are shown in the serving bucket; others are explored in the random bucket until their scores are better than the current best articles; at this point they get promoted to the serving bucket.

  In our serving bucket, we serve the same article at the F1 position in a given 5-minute window (except for overrides by business rules). Separate online models are tracked for articles at the F1 position in each bucket. Articles promoted to the F1 position in the serving bucket are subsequently scored by their model in the serving bucket; articles not playing at the F1 position in the serving bucket are of course scored by their model in the random bucket.

- **Explore/Exploit Strategy**: The random bucket is used for two purposes: (a) It provides a simple *explore-exploit* strategy that does random exploration with a small probability $P$,

and serves best articles ranked by their estimated scores from online models with a large probability $1 - P$. In addition, it helps us estimate systematic effects (e.g., diurnal CTR pattern) without the need to build elaborate statistical models that adjust for serving bias in other non-random buckets. Thus far, we have collected about 8 months of data from this continuously running random bucket; this has proved extremely useful in studying various offline models, running offline evaluations and conducting simulation studies.

The randomization procedure adopted is simple but proved effective in our setting. Our setting is different from the ones studied in classical explore-exploit literature; developing better strategies is a focus of our ongoing work. (See Section 6.)

## 4  Online Models

Tracking article CTR in an online fashion is a well studied area in time series with several methods available in the literature [3][7]; but the application to content optimization has not been carefully studied. We provide a description of three dynamic models that are currently used in our system.

### 4.1  Estimated Most Popular: EMP

This model tracks the log-odds of CTR per article at the F1 position over time but does not use any user features. The subscript $t$ in our notation refers to the $t^{th}$ interval after the article is first displayed in the bucket. Let $c_t$, $n_t$ denote the number of clicks and views at time $t$, we work with the the empirical logistic transform defined as $y_t = log(c_t + 0.5) - log(n_t - c_t + 0.5)$, approximately Gaussian for large $n_t$ with variance $w_t = (c_t + 0.5)^{-1} + (n_t - c_t + 0.5)^{-1}$ [6]. In our scenario, we get roughly $300 - 400$ observations at the F1 position per article in a 5-minute interval in the random bucket, hence the above transformation is appropriate for EMP and SS with few tens of user segments. Given that there may be a decay pattern in log-odds of CTR at the F1 position with increasing article lifetime, we fit a dynamic linear growth curve model which is given by

$$
\begin{aligned}
y_t &= o_t + \mu_t + \epsilon_t \sim N(0, V w_t) \\
\mu_t &= \mu_{t-1} + \beta_t + \delta \mu_t \sim N(0, \sigma^2_{\mu_t}) \\
\beta_t &= \beta_{t-1} + \delta \beta_t \sim N(0, \sigma^2_{\beta_t})
\end{aligned}
\tag{1}
$$

In Equation 1, $o_t$ is a constant offset term obtained from an offline model (e.g. hour-of-day correction), $\mu_t$ is the mean of $y_t$ at time $t$ and $\beta_t$ has the interpretation of *incremental decay* in the level of the series over the time interval from $t - 1$ to $t$, evolving during that interval according to the addition of the stochastic element $\delta \beta_t$. The evolution errors $\delta \mu_t$ and $\delta \beta_t$ are assumed to be uncorrelated. Model parameters are initialized by observing values at $t = 1$ for an article in random bucket, actual tracking begins at $t = 2$. In general, the initialization takes place through a feature based offline model built using retrospective data.

To provide more intuition on how the state parameters $\theta_t = (\mu_t, \beta_t)$ evolve, assume the evolutions $\delta \mu_t$ and $\delta \beta_t$ are zero at each time point. Then, $\mu_t = \mu_0 + t\beta_0$, a linear trend fitted to the $y_t$ values through weighted least squares. The addition of non-zero evolution makes this straight line dynamic and helps in tracking decay over time. In fact, the values of state evolution variance components $\sigma^2_{\mu_t}$ and $\sigma^2_{\beta_t}$ relative to noise variance $V w_t$ determine the amount of temporal smoothing that occurs for the model; large relative values smooth more by using a larger history to predict the future. Model fitting is conducted through a Kalman filter based on a *discounting* concept as explained in [7]. Details are discussed in [1].

### 4.2  Saturated Segmented Model: SS

This model generalizes EMP to incorporate user features. In particular, user covariates are used to create disjoint subsets (segments), a local EMP model is built to track item performance in each user segment. For a small number of user segments, we fit a separate EMP model per user segment for a given item at the F1 position. As the number of user segments grows, data sparseness may lead to high variance estimates in small segments, especially during early part of article lifetime. To address this, we smooth article scores in segments at each time point through a Bayesian hierarchical model.

In particular, if $(a_{it}, Q_{it}), i = 1, \cdots, k$, are predicted mean and variances of item score at F1 in $k$ different user segments at time $t$, we derive a new score as follows:

$$\tilde{a_{it}} = \frac{\tau}{\tau + Q_{it}} a_{it} + \frac{Q_{it}}{\tau + Q_{it}} \bar{a}_t \qquad (2)$$

where $\bar{a}_t$ is the EMP model score for the item. The constant $\tau$ that controls the amount of "shrinkage" towards the most popular is obtained by the DerSimonian and Laird estimator [10], widely used in meta-analysis.

### 4.3 Online Logistic Regression: OLR

The SS does not provide the flexibility to incorporate lower order interactions when working with a large number of features. For instance, given age, gender and geo-location for users, the SS model considers all possible combinations of the three variables. It is possible that an additive model based on two-factor interations (age-gender, age-geo, and gender-geo) may provide better performance. We use an efficient online logistic regression approach to build such models. The OLR updates parameter for every labelled event, positive or negative. Instead of achieving additivity by empirically transforming the data as in EMP and SS, it posits a Bernoulli likelihood for each event and achieves linearity by parametrizing the log-odds as a linear function of features. However, this makes the problem non-linear; we perform a quadratic approximation through a Taylor series expansion to achieve linearity. Modeling and fitting details are discussed in [1].

## 5 Experiments

In this section, we present the results of experiments that compare serving schemes based on our three online models (EMP, SS, and OLR) with the current editorial programming approach (which we refer to as ED). We show our online models significantly outperform ED based on bucket testsing the four alternatives concurrently on live traffic on the portal over a month. Then, by offline analysis, we identified the reason personalization (based on user features in OLR or segmentation in SS) did not provide improvement—it is mainly because we did not have sufficiently diverse articles to exploit, although SS and OLR are more predictive models than EMP. Finally, by extensive bucket tests on live traffic (which is an expensive and unusual opportunity for evaluating algorithms), we cast serious doubts on the usefulness of the common practice of comparing serving algorithms based on retrospective data (collected while using another serving scheme), and suggest that, without a random bucket or an effective correction procedure, it is *essential* to conduct tests on live traffic for statistical validity.

**Bucket Testing Methodology**: After conducting extensive offline analysis and several small pilots with different variations of models (including feature selection), we narrowed the candidates for live-traffic evaluation to the following: (1) EMP, (2) SS with Age $\times$ Gender segments, (3) OLR with features terms: Article + Position + Age$\times$ContentCategory + Gender$\times$ContentCategory (geo-location and user behavioral features were also bucket-tested in some other periods of time and provided no statistically significant improvement), and (4) ED. We used each of these schemes to serve traffic in a bucket (a random set of users) for one month; the four buckets ran concurrently. We measure the performance of a scheme by the *lifts* in terms of CTR relative to the baseline ED scheme. We also obtained significant improvements relative to round-robin serving scheme in the random bucket but do not report it to avoid clutter.

**Online Model Comparison**: Our online models significantly increased CTR over the original manual editorial scheme. Moreover, the increase in CTR was achieved mainly due to increased *reach*, i.e., we induced more users to click on articles. This provides evidence in favor of a strategy where various constraints in content programming are incorporated by human editors and algorithms are used to place them intelligently to optimize easily measurable metric like CTR. Figure 2 (a) shows the CTR lifts of different algorithms during one month. All three online models (EMP, SS and OLR) are significantly better than ED, with CTR lifts in the range of $30\% - 60\%$. This clearly demonstrates the ability of our online models to accurately track CTRs in real-time. Shockingly, the models that are based on user features, SS and OLR, are not statistically different from EMP, indicating that personalization to our selected user segments does not provide additional lift relative to EMP, although both SS and OLR have better predictive likelihoods relative to EMP on retrospective data analysis.

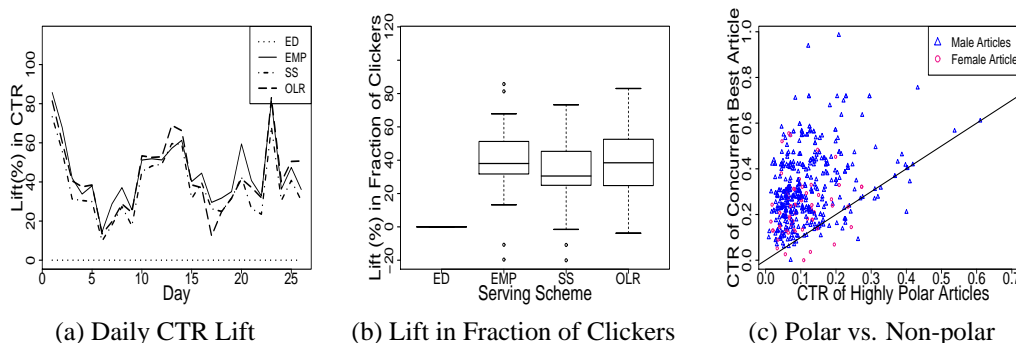

|  (a) Daily CTR Lift | (b) Lift in Fraction of Clickers | (c) Polar vs. Non-polar |

Figure 2: *Experimental results: (a) and (b) show bucket test results. (c) on the x-axis is the CTR of a polar article in a segment, on the y-axis is the CTR of the global best article (during the polar article's lifetime) in the same segment. Refer to text for definition of polar.*

Figure 2 (b) shows the lift relative to ED in terms of the fraction of clicking users. It shows that the lift achieved by the online models is not confined to a small cohort of users, but reflects conversion of more users to clickers.

**Analysis of Personalization**: We did not expect to find that personalization to user segments provided no additional CTR lift relative to EMP despite the fact that user features were predictive of article CTR. A closer look at the data revealed the main cause to be the current editorial content generation process, which is geared to create candidate articles that are expected to be popular for all users (not for different user segments). In fact, there *were* articles that have more affinity to some user segments than others—we define these to be articles whose CTR in a given segment was at least twice the article's overall CTR, and refer to them as *polar* articles. However, whenever polar articles were in the pool of candidate articles, there was usually a non-polar one in the pool that was more popular than the polar ones across all segments. As a result, we should chose the same non-polar one for all segments. Figure 2 (c) shows, for the gender segmentation, that polar articles almost always co-exist with an article whose overall CTR is greater than even the CTR in the segment of the polar article. For the AgeXGender segmentation, the global-best article was the same as the segment-best article about 65% of the intervals; the maximum expected CTR lift over global ranking was only about 1%. We observe similar patterns for segmentations based on other user features.

**Retrospective Evaluation Metrics vs. Bucket Tests**: It is common practice in existing literature to evaluate a new serving algorithm using predictive metrics obtained by running the algorithm on retrospective data (collected while using another serving scheme). For instance, such an approach has been used extensively in studying ad matching problems [11]. In our setting, this is equivalent to comparing a new serving scheme (e.g., EMP, SS, or OLR) to ED by computing some predictive metric on retrospective data obtained from ED. We found the performance differences obtained using retrospective data do not correlate well to those obtained by runing on live traffic [1]. This finding underscores the need for random bucket data, effective techniques to correct the bias, or a rapid bucket testing infrastructure to compare serving schemes.

## 6   Related Work

Google News personalization [13], which uses collaborative filtering to provide near real-time recommendation of news articles to users, is the most closely related prior work. However, while they select from a vast pool of unedited content aggregated from news sites across the globe, we recommend a subset from a small list of items chosen by editors. On the one hand, this allows us to build per-article models in near real-time; on the other, the editorially controlled mix of items means all articles are of high quality (making it hard to achieve lift by simply eliminating bad articles). Recent work on matching ads to queries [11] and ads to webpages [2] are related. However, their primary emphasis is on constructing accurate feature-based offline models that are updated at longer time intervals (e.g., daily), such models provide good initialization to our online models but perform poorly for reasons discussed in section 2.2. In [9], the authors consider an active exploration strategy to improve search rankings, which is similar in spirit to our randomization procedure. Our problem is also related to the rich literature on multi-armed bandit problems [5][8][14][12]. However, we note

that many of the assumptions made in the classical multi-armed bandit and reinforcement learning literature are not satisfied in our setting (dynamic set of articles, short article lifetime, batch-learning, non-stationary CTR, lagged response). In fact, short article lifetimes, dynamism of the content pool and the importance of learning article behaviour very quickly are the major challenges in our scenario. Preliminary experiments performed by obvious and natural modifications to the widely used UCB1 scheme [8] performed poorly. In a recent study [4] of a content aggregation site, digg.com, Wu et al. built a model for story popularity. However, their analysis is based on biased retrospective data, whereas we deployed our models and present results from tests conducted on live traffic.

## 7 Discussion

In this paper, we described *content optimization*, the problem of selecting articles to present to a user who is intent on browsing for information. There are many variants of the problem, depending on the setting. One variant is selecting from a very large and diverse pool of articles. Examples include recommending RSS feeds or articles from one or more RSS feeds, such as Google's news aggregation, and segmentation and personalization are likely to be effective. The variant that we addressed involves selecting from a small, homogeneous set of articles; segmentation may not be effective unless the pool of articles is chosen to be diverse, and there is a high premium in quickly estimating and tracking popularity per-article.

Our work suggests offline feature based models are not good enough to rank articles in a highly dynamic content publishing system where article pools are small, dynamic and of high quality; lifetimes are short; and the utility metric being measured (e.g., CTR) has a strong dynamic component. In fact, the biased nature of data obtained from a non-randomized serving scheme also underscores the need to obtain some percentage of data from a randomized experimental design. The delicate tradeoffs involved in maximizing utility (e.g., total number of clicks) by quickly converging to the best article for a given user (or user segment) through online models that are effectively initialized through offline feature based models (after adjusting for confounding factors), and performing unbiased exploration through small randomized experiments are the key machine learning challenges in this setting. While we have addressed them sufficiently well to handle small content pools, dealing with larger pools will require significant advances, and is the focus of our current research.

## Footnotes

[1]The data collected is anonymized, making it impossible to relate the activity to individual users.

## References

[1] D. Agarwal, B-C.Chen, P. Elango, and et al. Online models for content optimization, Yahoo! Technical Report TR-2008-004. 2008.

[2] D. Agarwal, A. Broder, D. Chakrabarti, D. Diklic, V. Josifovski, and M. Sayyadian. Estimating rates of rare events at multiple resolutions. In *KDD*, pages 16–25, New York, NY, USA, 2007. ACM.

[3] B. D. Anderson and J.B.Moore. *Optimal Filtering*. Dover, 1974.

[4] F.Wu and B.A.Huberman. Novelty and collective attention. 104:17599–17601, 2007.

[5] J.C.Gittins. Bandit processes and dynamic allocation indices. *Journal of the Royal Statistical Society, Series B*, 41:148–177, 1979.

[6] P. McCullagh and J. A. Nelder. *Generalized Linear Models*. Chapman & Hall/CRC, 1989.

[7] M.West and J.Harrison. *Bayesian Forecasting and Dynamic Models*. Springer-Verlag, 1997.

[8] P.Auer, N.Cesa-Bianchi, and P.Fischer. Finite-time analysis of the multiarmed bandit problem. *Machine Learning*, 47:235–256, 2002.

[9] F. Radlinski and T. Joachims. Active exploration for learning rankings from clickthrough data. In *ACM SIGKDD International Conference On Knowledge Discovery and Data Mining (KDD)*, 2007.

[10] R.DerSimonian and N.M.Laird. Meta-analysis in clinical trials. *Controlled Clinical Trials*, 7, 1986.

[11] M. Richardson, E. Dominowska, and R. Ragno. Predicting clicks: estimating the click-through rate for new ads. In *WWW*, pages 521–530, 2007.

[12] P. Sandeep, D. Agarwal, D. Chakrabarti, and V. Josifovski. Bandits for taxonomies: A model-based approach. In *In Proc. of the SIAM intl. conf. on Data Mining*, 2007.

[13] S.Das, D.Data, and A.Garg. Google news personalization:scalable online collaborative filtering. In *WWW, Banff, Alberta, Canada*, 2007.

[14] T.Lai and H.Robbins. Asymptotically efficient adaptive allocation rules. *Advances in Applied Mathematics*, 6:4–22, 1985.
